# Unsupervised Classification with Non-Gaussian Mixture Models using ICA

Te-Won Lee, Michael S. Lewicki and Terrence Sejnowski

Howard Hughes Medical Institute
Computational Neurobiology Laboratory
The Salk Institute
10010 N. Torrey Pines Road
La Jolla, California 92037, USA
{tewon,lewicki,terry}@salk.edu

## Abstract

We present an unsupervised classification algorithm based on an ICA mixture model. The ICA mixture model assumes that the observed data can be categorized into several mutually exclusive data classes in which the components in each class are generated by a linear mixture of independent sources. The algorithm finds the independent sources, the mixing matrix for each class and also computes the class membership probability for each data point. This approach extends the Gaussian mixture model so that the classes can have non-Gaussian structure. We demonstrate that this method can learn efficient codes to represent images of natural scenes and text. The learned classes of basis functions yield a better approximation of the underlying distributions of the data, and thus can provide greater coding efficiency. We believe that this method is well suited to modeling structure in high-dimensional data and has many potential applications.

## 1 Introduction

Recently, Blind Source Separation (BSS) by Independent Component Analysis (ICA) has shown promise in signal processing applications including speech enhancement systems, telecommunications and medical signal processing. ICA is a technique for finding a linear non-orthogonal coordinate system in multivariate data. The directions of the axes of this coordinate system are determined by the data's second- and higher-order statistics. The goal of the ICA is to linearly transform the data such that the transformed variables are as statistically independent from each

other as possible (Bell and Sejnowski, 1995; Cardoso and Laheld, 1996; Lee et al., 1999a). ICA generalizes the technique of Principal Component Analysis (PCA) and, like PCA, has proven a useful tool for finding structure in data.

One limitation of ICA is the assumption that the sources are independent. Here, we present an approach for relaxing this assumption using mixture models. In a mixture model (Duda and Hart, 1973), the observed data can be categorized into several mutually exclusive classes. When the class variables are modeled as multivariate Gaussian densities, it is called a Gaussian mixture model. We generalize the Gaussian mixture model by modeling each class with independent variables (ICA mixture model). This allows modeling of classes with non-Gaussian (e.g., platykurtic or leptokurtic) structure. An algorithm for learning the parameters is derived using the expectation maximization (EM) algorithm. In Lee et al. (1999c), we demonstrated that this approach showed improved performance in data classification problems. Here, we apply the algorithm to learning efficient codes for representing different types of images.

## 2 The ICA Mixture Model

We assume that the data were generated by a mixture density (Duda and Hart, 1973):

$$p(\mathbf{x}|\Theta) = \sum_{k=1}^{K} p(\mathbf{x}|C_k, \theta_k) p(C_k), \qquad (1)$$

where $\Theta = (\theta_1, \cdots, \theta_K)$ are the unknown parameters for each $p(\mathbf{x}|C_k, \theta_k)$, called the component densities. We further assume that the number of classes, $K$, and the a priori probability, $p(C_k)$, for each class are known. In the case of a Gaussian mixture model, $p(\mathbf{x}|C_k, \theta_k) \propto N(\mu_k, \Sigma_k)$. Here we assume that the form of the component densities is non-Gaussian and the data within each class are described by an ICA model.

$$\mathbf{x}_k = \mathbf{A}_k \mathbf{s}_k + \mathbf{b}_k, \qquad (2)$$

where $\mathbf{A}_k$ is a $N \times M$ scalar matrix (called the basis or mixing matrix) and $\mathbf{b}_k$ is the bias vector for class $k$. The vector $\mathbf{s}_k$ is called the source vector (these are also the coefficients for each basis vector). It is assumed that the individual sources $s_i$ within each class are mutually independent across a data ensemble. For simplicity, we consider the case where $\mathbf{A}_k$ is full rank, i.e. the number of sources ($M$) is equal to the number of mixtures ($N$). Figure 1 shows a simple example of a dataset that can be described by ICA mixture model. Each class was generated from eq.2 using a different $\mathbf{A}$ and $\mathbf{b}$. Class (o) was generated by two uniform distributed sources, whereas class (+) was generated by two Laplacian distributed sources ($p(s) \propto \exp(-|s|)$). The task is to model the unlabeled data points and to determine the parameters for each class, $\mathbf{A}_k, \mathbf{b}_k$ and the probability of each class $p(C_k|\mathbf{x}, \theta_{1:K})$ for each data point. A learning algorithm can be derived by an expectation maximization approach (Ghahramani, 1994) and implemented in the following steps:

- Compute the log-likelihood of the data for each class:
$$\log p(\mathbf{x}|C_k, \theta_k) = \log p(\mathbf{s}_k) - \log(\det |\mathbf{A}_k|), \qquad (3)$$
where $\theta_k = \{\mathbf{A}_k, \mathbf{b}_k, \mathbf{s}_k\}$.

- Compute the probability for each class given the data vector $\mathbf{x}$
$$p(C_k|\mathbf{x}, \theta_{1:K}) = \frac{p(\mathbf{x}|\theta_k, C_k) p(C_k)}{\sum_k p(\mathbf{x}|\theta_k, C_k) p(C_k)}. \qquad (4)$$

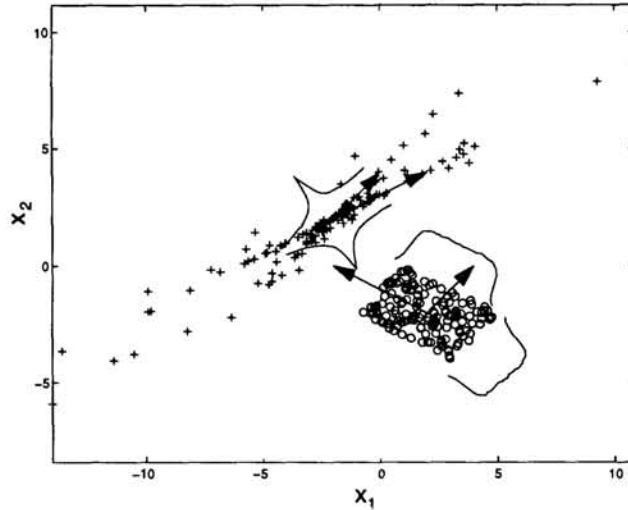

Figure 1: A simple example for classifying an ICA mixture model. There are
two classes (+) and (o); each class was generated by two independent variables,
two bias terms and two basis vectors. Class (o) was generated by two uniform
distributed sources as indicated next to the data class. Class (+) was generated by
two Laplacian distributed sources with a sharp peak at the bias and heavy tails.
The inset graphs show the distributions of the source variables, $\mathbf{s}_{i,k}$, for each basis
vector.

- Adapt the basis functions $\mathbf{A}$ and the bias terms $\mathbf{b}$ for each class. The basis
  functions are adapted using gradient ascent

$$
\begin{aligned}
\Delta\mathbf{A}_k \quad &\propto \quad \frac{\partial}{\partial\mathbf{A}_k}\log p(\mathbf{x}|\theta_{1:K}) \\
&= \quad p(C_k|\mathbf{x},\theta_{1:K})\frac{\partial}{\partial\mathbf{A}_k}\log p(\mathbf{x}|C_k,\theta_k).
\end{aligned} \tag{5}
$$

Note that this simply weights any standard ICA algorithm gradient by
$p(C_k|\mathbf{x},\theta_{1:K})$. The gradient can also be summed over multiple data points.
The bias term is updated according to

$$
\mathbf{b}_k = \frac{\sum_t \mathbf{x}_t p(C_k|\mathbf{x}_t,\theta_{1:K})}{\sum_t p(C_k|\mathbf{x}_t,\theta_{1:K})}, \tag{6}
$$

where $t$ is the data index ($t = 1,\ldots,T$).

The three steps in the learning algorithm perform gradient ascent on the total
likelihood of the data in eq.1.

The extended infomax ICA learning rule is able to blindly separate mixed sources
with sub- and super-Gaussian distributions. This is achieved by using a simple
type of learning rule first derived by Girolami (1998). The learning rule in Lee
et al. (1999b) uses the stability analysis of Cardoso and Laheld (1996) to switch
between sub- and super-Gaussian regimes. The learning rule expressed in terms of
$\mathbf{W} = \mathbf{A}^{-1}$, called the filter matrix is:

$$
\Delta\mathbf{W} \propto \left[\mathbf{I} - \mathbf{K}\tanh(\mathbf{u})\mathbf{u}^T - \mathbf{u}\mathbf{u}^T\right]\mathbf{W}, \tag{7}
$$

where $k_i$ are elements of the N-dimensional diagonal matrix $\mathbf{K}$ and $\mathbf{u} = \mathbf{W}\mathbf{x}$. The unmixed sources $\mathbf{u}$ are the source estimate $\mathbf{s}$ (Bell and Sejnowski, 1995). The $k_i$'s are (Lee et al., 1999b)

$$k_i = \mathrm{sign}\left(E[\mathrm{sech}^2 u_i]E[u_i^2] - E[u_i \tanh u_i]\right). \tag{8}$$

The source distribution is super-Gaussian when $k_i = 1$ and sub-Gaussian when $k_i = -1$. For the log-likelihood estimation in eq.3 the term $\log p(\mathbf{s})$ can be approximated as follows

$$\log p(\mathbf{s}) \propto - \sum_n \log \cosh s_n - \frac{s_n^2}{2} \qquad \text{super-Gaussian}$$

$$\log p(\mathbf{s}) \propto + \sum_n \log \cosh s_n - \frac{s_n^2}{2} \qquad \text{sub-Gaussian} \tag{9}$$

Super-Gaussian densities, are approximated by a density model with heavier tail than the Gaussian density; Sub-Gaussian densities are approximated by a bimodal density (Girolami, 1998). Although the source density approximation is crude it has been demonstrated that they are sufficient for standard ICA problems (Lee et al., 1999b). When learning sparse representations only, a Laplacian prior ($p(s) \propto \exp(-|s|)$) can be used for the weight update which simplifies the infomax learning rule to

$$\Delta \mathbf{W} \quad \propto \quad [\mathbf{I} - \mathrm{sign}(\mathbf{u})\mathbf{u}^T]\,\mathbf{W}, \tag{10}$$

$$\log p(\mathbf{s}) \quad \propto \quad - \sum_n |s_n| \qquad \text{Laplacian prior}$$

## 3    Learning efficient codes for images

Recently, several approaches have been proposed to learn image codes that utilize a set of linear basis functions. Olshausen and Field (1996) used a sparseness criterion and found codes that were similar to localized and oriented receptive fields. Similar results were presented by Bell and Sejnowski (1997) using the infomax algorithm and by Lewicki and Olshausen (1998) using a Bayesian approach. By applying the ICA mixture model we present results which show a higher degree of flexibility in encoding the images. We used images of natural scenes obtained from Olshausen and Field (1996) and text images of scanned newspaper articles. The training set consisted of 12 by 12 pixel patches selected randomly from both image types. Figure 2 illustrates examples of those image patches. Two complete basis vectors $\mathbf{A}_1$ and $\mathbf{A}_2$ were randomly initialized. Then, for each gradient in eq.5 a stepsize was computed as a function of the amplitude of the basis vectors and the number of iterations. The algorithm converged after 100,000 iterations and learned two classes of basis functions as shown in figure 3. Figure 3 (top) shows basis functions corresponding to natural images. The basis functions show Gabor[1]-like structure as previously reported in (Olshausen and Field, 1996; Bell and Sejnowski, 1997; Lewicki and Olshausen, 1998). However, figure 3 (bottom) shows basis functions corresponding to text images. These basis functions resemble bars with different lengths and widths that capture the high-frequency structure present in the text images.

### 3.1    Comparing coding efficiency

We have compared the coding efficiency between the ICA mixture model and similar models using Shannon's theorem to obtain a lower bound on the number of bits

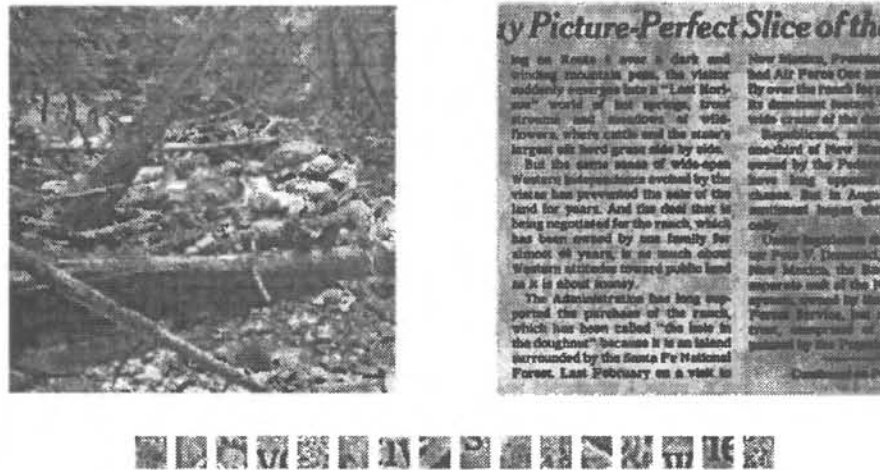

Figure 2: Example of natural scene and text image. The 12 by 12 pixel image patches were randomly sampled from the images and used as inputs to the ICA mixture model.

required to encode the pattern.

$$\#\text{bits} \geq -\log_2 P(\mathbf{x}|\mathbf{A}) - N\log_2(\sigma_x), \tag{11}$$

where $N$ is the dimensionality of the input pattern $\mathbf{x}$ and $\sigma_x$ is the coding precision (standard deviation of the noise introduced by errors in encoding). Table 1 compares the coding efficiency of five different methods. It shows the number of bits required to encode three different test data sets (5000 image patches from natural scenes, 5000 image patches from text images and 5000 image patches from both image types) using five different encoding methods (ICA mixture model, nature trained ICA, text trained ICA, nature and text trained ICA, and PCA trained on all three test sets). It is clear that ICA basis functions trained on natural scene images exhibit the best encoding when only natural scenes are presented (column: nature). The same applies to text images (column: text). Note that text training yields a reasonable basis for both data sets but nature training gives a good basis only for nature. The ICA mixture model shows the same encoding power for the individual test data sets, and it gives the best encoding when both image types are present. In this case, the encoding difference between the ICA mixture model and PCA is significant (more than 20%). ICA mixtures yielded a small improvement over ICA trained on both image types. We expect the size of the improvement to be greater in situations where there are greater differences among the classes. An advantage of the mixture model is that each image patch is automatically classified. ,

## 4   Discussion

The new algorithm for unsupervised classification presented here is based on a maximum likelihood mixture model using ICA to model the structure of the classes. We have demonstrated here that the algorithm can learn efficient codes to represent different image types such as natural scenes and text images. In this case, the learned classes of basis functions show a 20% improvement over PCA encoding. ICA mixture model should show better image compression rates than traditional compression algorithm such as JPEG.

The ICA mixture model is a nonlinear model in which each class is modeled as a linear process and the choice of class is modeled using probabilities. This model

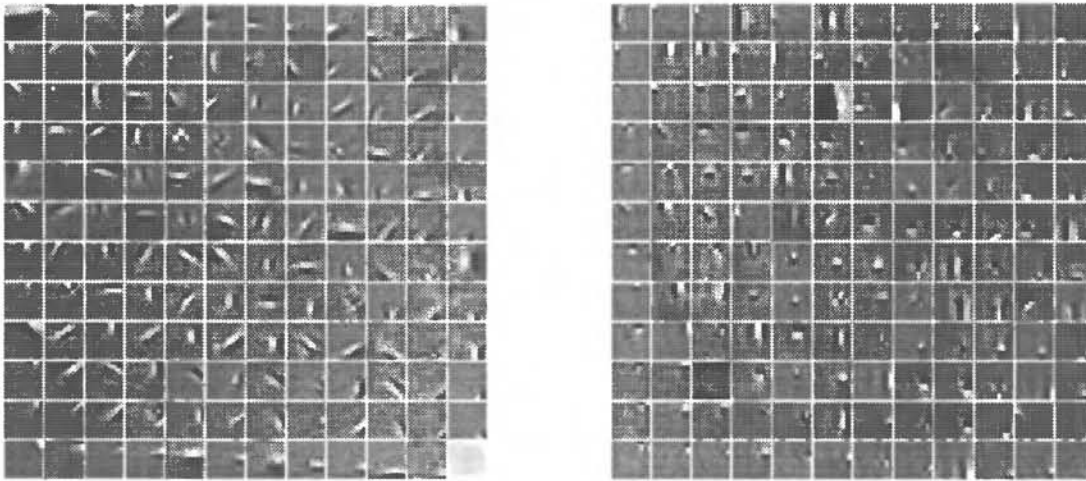

Figure 3: (Left) Basis function class corresponding to natural images. (Right) Basis function class corresponding to text images.

Table 1: Comparing coding efficiency

| Training set and model | Test data | | |
|---|---|---|---|
| | Nature | Text | Nature and Text |
| ICA mixtures | 4.72 | 5.20 | 4.96 |
| Nature trained ICA | 4.72 | 9.57 | 7.15 |
| Text trained ICA | 5.00 | 5.19 | 5.10 |
| Nature and text trained ICA | 4.83 | 5.29 | 5.07 |
| PCA | 6.22 | 5.97 | 6.09 |

Coding efficiency (bits per pixel) of five methods is compared for three test sets. Coding precision was set to 7 bits (Nature: $\sigma_x = 0.016$ and Text: $\sigma_x = 0.029$).

can therefore be seen as a nonlinear ICA model. Furthermore, it is one way of relaxing the independence assumption over the whole data set. The ICA mixture model is a conditional independence model, i.e., the independence assumption holds within only each class and there may be dependencies among classes. A different view of the ICA mixture model is to think of the classes of being an overcomplete representation. Compared to the approach of Lewicki and Sejnowski (1998), the main difference is that the basis functions learned here are mutually exclusive, i.e. each class uses its own set of basis functions.

This method is similar to other approaches including the mixture density networks by Bishop (1994) in which a neural network was used to find arbitrary density functions. This algorithm reduces to the Gaussian mixture model when the source priors are Gaussian. Purely Gaussian structure, however, is rare in real data sets. Here we have used priors of the form of super-Gaussian and sub-Gaussian densities. But these could be extended as proposed by Attias (1999). The proposed model was used for learning a complete set of basis functions without additive noise. However, the method can be extended to take into account additive Gaussian noise and an overcomplete set of basis vectors (Lewicki and Sejnowski, 1998).

In (Lee et al., 1999c), we have performed several experiments on benchmark data sets for classification problems. The results were comparable or improved over those obtained by AutoClass (Stutz and Cheeseman, 1994) which uses a Gaussian mixture

model. Furthermore, we showed that the algorithm can be applied to blind source separation in nonstationary environments. The method can switch automatically between learned mixing matrices in different environments (Lee et al., 1999c). This may prove to be useful in the automatic detection of sleep stages by observing EEG signals. The method can identify these stages due to the changing source priors and their mixing.

Potential applications of the proposed method include the problem of noise removal and the problem of filling in missing pixels. We believe that this method provides greater flexibility in modeling structure in high-dimensional data and has many potential applications.

## Footnotes

[1]Gaussian modulated siusoidal

# References

Attias, H. (1999). Blind separation of noisy mixtures: An EM algorithm for independent factor analysis. *Neural Computation*, in press.

Bell, A. J. and Sejnowski, T. J. (1995). An Information-Maximization Approach to Blind Separation and Blind Deconvolution. *Neural Computation*, 7:1129–1159.

Bell, A. J. and Sejnowski, T. J. (1997). The 'independent components' of natural scenes are edge filters. *Vision Research*, 37(23):3327–3338.

Bishop, C. (1994). Mixture density networks. *Technical Report*, NCRG/4288.

Cardoso, J.-F. and Laheld, B. (1996). Equivariant adaptive source separation. *IEEE Trans. on S.P.*, 45(2):434–444.

Duda, R. and Hart, P. (1973). *Pattern classification and scene analysis*. Wiley, New York.

Ghahramani, Z. (1994). Solving inverse problems using an em approach to density estimation. *Proceedings of the 1993 Connectionist Models Summer School*, pages 316–323.

Girolami, M. (1998). An alternative perspective on adaptive independent component analysis algorithms. *Neural Computation*, 10(8):2103–2114.

Lee, T.-W., Girolami, M., Bell, A. J., and Sejnowski, T. J. (1999a). A unifying framework for independent component analysis. *International Journal on Mathematical and Computer Models*, in press.

Lee, T.-W., Girolami, M., and Sejnowski, T. J. (1999b). Independent component analysis using an extended infomax algorithm for mixed sub-gaussian and super-gaussian sources. *Neural Computation*, 11(2):409–433.

Lee, T.-W., Lewicki, M. S., and Sejnowski, T. J. (1999c). ICA mixture models for unsupervised classification and automatic context switching. In *International Workshop on ICA*, Aussois, in press.

Lewicki, M. and Olshausen, B. (1998). Inferring sparse, overcomplete image codes using an efficient coding framework. In *Advances in Neural Information Processing Systems 10*, pages 556–562.

Lewicki, M. and Sejnowski, T. J. (1998). Learning nonlinear overcomplete representations for efficient coding. In *Advances in Neural Information Processing Systems 10*, pages 815–821.

Olshausen, B. and Field, D. (1996). Emergence of simple-cell receptive field properties by learning a sparse code for natural images. *Nature*, 381:607–609.

Stutz, J. and Cheeseman, P. (1994). Autoclass - a Bayesian approach to classification. *Maximum Entropy and Bayesian Methods*, Kluwer Academic Publishers.
